# Sparse Convolved Gaussian Processes for Multi-output Regression

**Mauricio Alvarez**
School of Computer Science
University of Manchester, U.K.
`alvarezm@cs.man.ac.uk`

**Neil D. Lawrence**
School of Computer Science
University of Manchester, U.K.
`neill@cs.man.ac.uk`

## Abstract

We present a sparse approximation approach for dependent output Gaussian processes (GP). Employing a latent function framework, we apply the convolution process formalism to establish dependencies between output variables, where each latent function is represented as a GP. Based on these latent functions, we establish an approximation scheme using a conditional independence assumption between the output processes, leading to an approximation of the full covariance which is determined by the locations at which the latent functions are evaluated. We show results of the proposed methodology for synthetic data and real world applications on pollution prediction and a sensor network.

## 1 Introduction

We consider the problem of modeling correlated outputs from a single Gaussian process (GP). Applications of modeling multiple outputs include multi-task learning (see *e.g.* [1]) and jointly predicting the concentration of different heavy metal pollutants [5]. Modelling multiple output variables is a challenge as we are required to compute cross covariances between the different outputs. In geostatistics this is known as *cokriging*. Whilst cross covariances allow us to improve our predictions of one output given the others because the correlations between outputs are modelled [6, 2, 15, 12] they also come with a computational and storage overhead. The main aim of this paper is to address these overheads in the context of convolution processes [6, 2].

One neat approach to account for non-trivial correlations between outputs employs convolution processes (CP). When using CPs each output can be expressed as the convolution between a smoothing kernel and a *latent* function [6, 2]. Let's assume that the latent function is drawn from a GP. If we also share the same latent function across several convolutions (each with a potentially different smoothing kernel) then, since a convolution is a linear operator on a function, the outputs of the convolutions can be expressed as a jointly distributed GP. It is this GP that is used to model the multi-output regression. This approach was proposed by [6, 2] who focussed on a white noise process for the latent function.

Even though the CP framework is an elegant way for constructing dependent output processes, the fact that the full covariance function of the joint GP must be considered results in significant storage and computational demands. For $Q$ output dimensions and $N$ data points the covariance matrix scales as $QN$ leading to $O(Q^3N^3)$ computational complexity and $O(N^2Q^2)$ storage. Whilst other approaches to modeling multiple output regression are typically more constraining in the types of cross covariance that can be expressed [1, 15], these constraints also lead to structured covariances functions for which inference and learning are typically more efficient (typically for $N > Q$ these methods have $O(N^3Q)$ computation and $O(N^2Q)$ storage). We are interested in exploiting the richer class of covariance structures allowed by the CP framework, but without the additional computational overhead they imply.

We propose a sparse approximation for the full covariance matrix involved in the multiple output convolution process, exploiting the fact that each of the outputs is conditional independent of all others given the input process. This leads to an approximation for the covariance matrix which keeps intact the covariances of each output and approximates the cross-covariances terms with a low rank matrix. Inference and learning can then be undertaken with the same computational complexity as a set of independent GPs. The approximation turns out to be strongly related to the partially independent training conditional (PITC) [10] approximation for a single output GP. This inspires us to consider a further conditional independence function across data points that leads to an approximation which shares the form of the fully independent training conditional (FITC) approximation [13, 10] reducing computational complexity to $O(NQM^2)$ and storage to $O(NQM)$ with $M$ representing a user specified value.

To introduce our sparse approximation some review of the CP framework is required (Section 2). Then in Section 3, we present sparse approximations for the multi-output GP. We discuss relations with other approaches in Section 4. Finally, in Section 5, we demonstrate the approach on both synthetic and real datasets.

## 2 Convolution Processes

Consider a set of $Q$ functions $\{f_q(\mathbf{x})\}_{q=1}^Q$, where each function is expressed as the convolution between a smoothing kernel $\{k_q(\mathbf{x})\}_{q=1}^Q$, and a latent function $u(\mathbf{z})$,

$$f_q(\mathbf{x}) = \int_{-\infty}^{\infty} k_q(\mathbf{x} - \mathbf{z})u(\mathbf{z})\mathrm{d}\mathbf{z}.$$

More generally, we can consider the influence of more than one latent function, $\{u_r(\mathbf{z})\}_{r=1}^R$, and corrupt each of the outputs of the convolutions with an independent process (which could also include a noise term), $w_q(\mathbf{x})$, to obtain

$$y_q(\mathbf{x}) = f_q(\mathbf{x}) + w_q(\mathbf{x}) = \sum_{r=1}^R \int_{-\infty}^{\infty} k_{qr}(\mathbf{x} - \mathbf{z})u_r(\mathbf{z})\mathrm{d}\mathbf{z} + w_q(\mathbf{x}). \tag{1}$$

The covariance between two different functions $y_q(\mathbf{x})$ and $y_s(\mathbf{x}')$ is then recovered as

$$\mathrm{cov}\left[y_q(\mathbf{x}), y_s(\mathbf{x}')\right] = \mathrm{cov}\left[f_q(\mathbf{x}), f_s(\mathbf{x}')\right] + \mathrm{cov}\left[w_q(\mathbf{x}), w_s(\mathbf{x}')\right]\delta_{qs},$$

where

$$\mathrm{cov}\left[f_q(\mathbf{x}), f_s(\mathbf{x}')\right] = \sum_{r=1}^R \sum_{p=1}^R \int_{-\infty}^{\infty} k_{qr}(\mathbf{x} - \mathbf{z}) \int_{-\infty}^{\infty} k_{sp}(\mathbf{x}' - \mathbf{z}')\,\mathrm{cov}\left[u_r(\mathbf{z}), u_p(\mathbf{z}')\right]\mathrm{d}\mathbf{z}'\mathrm{d}\mathbf{z} \tag{2}$$

This equation is a general result; in [6, 2] the latent functions $u_r(\mathbf{z})$ are assumed as independent white Gaussian noise processes, *i.e.* $\mathrm{cov}\left[u_r(\mathbf{z}), u_p(\mathbf{z}')\right] = \sigma_{u_r}^2 \delta_{rp}\delta_{\mathbf{z},\mathbf{z}'}$, so the expression (2) is simplified as

$$\mathrm{cov}\left[f_q(\mathbf{x}), f_s(\mathbf{x}')\right] = \sum_{r=1}^R \sigma_{u_r}^2 \int_{-\infty}^{\infty} k_{qr}(\mathbf{x} - \mathbf{z})k_{sr}(\mathbf{x}' - \mathbf{z})\mathrm{d}\mathbf{z}.$$

We are going to relax this constraint on the latent processes, we assume that each inducing function is an independent GP, *i.e.* $\mathrm{cov}\left[u_r(\mathbf{z}), u_p(\mathbf{z}')\right] = k_{u_r u_p}(\mathbf{z}, \mathbf{z}')\delta_{rp}$, where $k_{u_r u_r}(\mathbf{z}, \mathbf{z}')$ is the covariance function for $u_r(\mathbf{z})$. With this simplification, (2) can be written as

$$\mathrm{cov}\left[f_q(\mathbf{x}), f_s(\mathbf{x}')\right] = \sum_{r=1}^R \int_{-\infty}^{\infty} k_{qr}(\mathbf{x} - \mathbf{z}) \int_{-\infty}^{\infty} k_{sr}(\mathbf{x}' - \mathbf{z}')k_{u_r u_r}(\mathbf{z}, \mathbf{z}')\mathrm{d}\mathbf{z}'\mathrm{d}\mathbf{z}. \tag{3}$$

As well as this correlation across outputs, the correlation between the latent function, $u_r(\mathbf{z})$, and any given output, $f_q(\mathbf{x})$, can be computed,

$$\mathrm{cov}\left[f_q(\mathbf{x}), u_r(\mathbf{z})\right)] = \int_{-\infty}^{\infty} k_{qr}(\mathbf{x} - \mathbf{z}')k_{u_r u_r}(\mathbf{z}', \mathbf{z})\mathrm{d}\mathbf{z}'. \tag{4}$$

## 3 Sparse Approximation

Given the convolution formalism, we can construct a full GP over the set of outputs. The likelihood of the model is given by

$$p(\mathbf{y}|\mathbf{X}, \boldsymbol{\phi}) = \mathcal{N}(\mathbf{0}, \mathbf{K_{f,f}} + \boldsymbol{\Sigma}), \tag{5}$$

where $\mathbf{y} = \left[\mathbf{y}_1^\top, \ldots, \mathbf{y}_Q^\top\right]^\top$ is the set of output functions with $\mathbf{y}_q = [y_q(\mathbf{x}_1), \ldots, y_q(\mathbf{x}_N)]^\top$; $\mathbf{K_{f,f}} \in \Re^{QN \times QN}$ is the covariance matrix relating all data points at all outputs, with elements $\text{cov}\left[f_q(\mathbf{x}), f_s(\mathbf{x}')\right]$ in (3); $\boldsymbol{\Sigma} = \Sigma \otimes \mathbf{I}_N$, where $\Sigma$ is a diagonal matrix with elements $\{\sigma_q^2\}_{q=1}^Q$; $\boldsymbol{\phi}$ is the set of parameters of the covariance matrix and $\mathbf{X} = \{\mathbf{x}_1, \ldots, \mathbf{x}_N\}$ is the set of training input vectors at which the covariance is evaluated.

The predictive distribution for a new set of input vectors $\mathbf{X}_*$ is [11]

$$p(\mathbf{y}_*|\mathbf{y}, \mathbf{X}, \mathbf{X}_*, \boldsymbol{\phi}) = \mathcal{N}\left(\mathbf{K_{f_*,f}}(\mathbf{K_{f,f}} + \boldsymbol{\Sigma})^{-1}\mathbf{y}, \mathbf{K_{f_*,f_*}} - \mathbf{K_{f_*,f}}(\mathbf{K_{f,f}} + \boldsymbol{\Sigma})^{-1}\mathbf{K_{f,f_*}} + \boldsymbol{\Sigma}\right),$$

where we have used $\mathbf{K_{f_*,f_*}}$ as a compact notation to indicate when the covariance matrix is evaluated at the inputs $\mathbf{X}_*$, with a similar notation for $\mathbf{K_{f_*,f}}$. Learning from the log-likelihood involves the computation of the inverse of $\mathbf{K_{f,f}} + \boldsymbol{\Sigma}$, which grows with complexity $\mathcal{O}((NQ)^3)$. Once the parameters have been learned, prediction is $\mathcal{O}(NQ)$ for the predictive mean and $\mathcal{O}((NQ)^2)$ for the predictive variance.

Our strategy for approximate inference is to exploit the natural conditional dependencies in the model. If we had observed the entire length of each latent function, $u_r(\mathbf{z})$, then from (1) we see that each $y_q(\mathbf{x})$ *would* be independent, *i.e.* we can write,

$$p(\{y_q(\mathbf{x})\}_{q=1}^Q \,|\, \{u_r(\mathbf{z})\}_{r=1}^R, \boldsymbol{\theta}) = \prod_{q=1}^Q p(y_q(\mathbf{x}) \,|\, \{u_r(\mathbf{z})\}_{r=1}^R, \boldsymbol{\theta}),$$

where $\boldsymbol{\theta}$ are the parameters of the kernels and covariance functions. Our key assumption is that this independence will hold even if we have only observed $M$ samples from $u_r(\mathbf{z})$ rather than the whole function. The observed values of these $M$ samples are then marginalized (as they are for the exact case) to obtain the approximation to the likelihood. Our intuition is that the approximation should be more accurate for larger $M$ and smoother latent functions, as in this domain the latent function could be very well characterized from only a few samples.

We define $\mathbf{u} = \left[\mathbf{u}_1^\top, \ldots, \mathbf{u}_R^\top\right]^\top$ as the samples from the latent function with $\mathbf{u}_r = [u_r(\mathbf{z}_1), \ldots, u_r(\mathbf{z}_M)]^\top$; $\mathbf{K_{u,u}}$ is then the covariance matrix between the samples from the latent functions $u_r(\mathbf{z})$, with elements given by $k_{u_r u_r}(\mathbf{z}, \mathbf{z}')$; $\mathbf{K_{f,u}} = \mathbf{K_{u,f}}^\top$ are the cross-covariance matrices between the latent functions $u_r(\mathbf{z})$ and the outputs $f_q(\mathbf{x})$, with elements $\text{cov}\left[f_q(\mathbf{x}), u_r(\mathbf{z})\right]$ in (4) and $\mathbf{Z} = \{\mathbf{z}_1, \ldots, \mathbf{z}_M\}$ is the set of input vectors at which the covariance $\mathbf{K_{u,u}}$ is evaluated.

We now make the conditional independence assumption given the samples from the latent functions,

$$p(\mathbf{y}|\mathbf{u}, \mathbf{Z}, \mathbf{X}, \boldsymbol{\theta}) = \prod_{q=1}^Q p(\mathbf{y}_q|\mathbf{u}, \mathbf{Z}, \mathbf{X}, \boldsymbol{\theta}) = \prod_{q=1}^Q \mathcal{N}\left(\mathbf{K_{f_q,u}}\mathbf{K_{u,u}^{-1}}\mathbf{u}, \mathbf{K_{f_q,f_q}} - \mathbf{K_{f_q,u}}\mathbf{K_{u,u}^{-1}}\mathbf{K_{u,f_q}} + \sigma_q^2\mathbf{I}\right).$$

We rewrite this product as a single Gaussian with a block diagonal covariance matrix,

$$p(\mathbf{y}|\mathbf{u}, \mathbf{Z}, \mathbf{X}, \boldsymbol{\theta}) = \mathcal{N}\left(\mathbf{K_{f,u}}\mathbf{K_{u,u}^{-1}}\mathbf{u}, \mathbf{D} + \boldsymbol{\Sigma}\right) \tag{6}$$

where $\mathbf{D} = \text{blockdiag}\left[\mathbf{K_{f,f}} - \mathbf{K_{f,u}}\mathbf{K_{u,u}^{-1}}\mathbf{K_{u,f}}\right]$, and we have used the notation $\text{blockdiag}[\mathbf{G}]$ to indicate the block associated with each output of the matrix $\mathbf{G}$ should be retained, but all other elements should be set to zero. We can also write this as $\mathbf{D} = \left[\mathbf{K_{f,f}} - \mathbf{K_{f,u}}\mathbf{K_{u,u}^{-1}}\mathbf{K_{u,f}}\right] \odot \mathbf{M}$ where $\odot$ is the Hadamard product and $\mathbf{M} = \mathbf{I}_Q \otimes \mathbf{1}_N$, $\mathbf{1}_N$ being the $N \times N$ matrix of ones and $\otimes$ being the Kronecker product. We now marginalize the values of the samples from the latent functions by using their process priors, *i.e.* $p(\mathbf{u}|\mathbf{Z}) = \mathcal{N}(\mathbf{0}, \mathbf{K_{u,u}})$. This leads to the following marginal likelihood,

$$p(\mathbf{y}|\mathbf{Z}, \mathbf{X}, \boldsymbol{\theta}) = \int p(\mathbf{y}|\mathbf{u}, \mathbf{Z}, \mathbf{X}, \boldsymbol{\theta})p(\mathbf{u}|\mathbf{Z})\mathrm{d}\mathbf{u} = \mathcal{N}\left(\mathbf{0}, \mathbf{D} + \mathbf{K_{f,u}}\mathbf{K_{u,u}^{-1}}\mathbf{K_{u,f}} + \boldsymbol{\Sigma}\right). \tag{7}$$

Notice that, compared to (5), the full covariance matrix $\mathbf{K_{f,f}}$ has been replaced by the low rank covariance $\mathbf{K_{f,u}K_{u,u}^{-1}K_{u,f}}$ in all entries except in the diagonal blocks corresponding to $\mathbf{K_{f_q,f_q}}$. When using the marginal likelihood for learning, the computation load is associated to the calculation of the inverse of $\mathbf{D}$. The complexity of this inversion is $\mathcal{O}(N^3Q) + O(NQM^2)$, storage of the matrix is $O(N^2Q) + O(NQM)$. Note that if we set $M = N$ these reduce to $\mathcal{O}(N^3Q)$ and $O(N^2Q)$ respectively which matches the computational complexity of applying $Q$ independent GPs to model the multiple outputs.

Combining eq. (6) with $p(\mathbf{u}|\mathbf{Z})$ using Bayes theorem, the posterior distribution over $\mathbf{u}$ is obtained as

$$p(\mathbf{u}|\mathbf{y}, \mathbf{X}, \mathbf{Z}, \boldsymbol{\theta}) = \mathcal{N}\left(\mathbf{K_{u,u}A^{-1}K_{u,f}}(\mathbf{D} + \boldsymbol{\Sigma})^{-1}\mathbf{y}, \mathbf{K_{u,u}A^{-1}K_{u,u}}\right) \qquad (8)$$

where $\mathbf{A} = \mathbf{K_{u,u}} + \mathbf{K_{u,f}}(\mathbf{D} + \boldsymbol{\Sigma})^{-1}\mathbf{K_{f,u}}$. The predictive distribution is expressed through the integration of (6), evaluated at $\mathbf{X_*}$, with (8), giving

$$p(\mathbf{y_*}|\mathbf{y}, \mathbf{X}, \mathbf{X_*}, \mathbf{Z}, \boldsymbol{\theta}) = \int p(\mathbf{y_*}|\mathbf{u}, \mathbf{Z}, \mathbf{X_*}, \boldsymbol{\theta})p(\mathbf{u}|\mathbf{y}, \mathbf{X}, \mathbf{Z}, \boldsymbol{\theta})\mathrm{d}\mathbf{u}$$
$$= \mathcal{N}\left(\mathbf{K_{f_*,u}A^{-1}K_{u,f}}(\mathbf{D} + \boldsymbol{\Sigma})^{-1}\mathbf{y}, \mathbf{D_*} + \mathbf{K_{f_*,u}A^{-1}K_{u,f_*}} + \boldsymbol{\Sigma}\right) \qquad (9)$$

with $\mathbf{D_*} = \mathrm{blockdiag}\left[\mathbf{K_{f_*,f_*}} - \mathbf{K_{f_*,u}K_{u,u}^{-1}K_{u,f_*}}\right]$.

The functional form of (7) is almost identical to that of the PITC approximation [10], with the samples we retain from the latent function providing the same role as the *inducing values* in the partially independent training conditional (PITC) approximation. This is perhaps not surprising given that the nature of the conditional independence assumptions in PITC is similar to that we have made. A key difference is that in PITC it is not obvious which variables should be grouped together when making the conditional independence assumption, here it is clear from the structure of the model that each of the outputs should be grouped separately. However, the similarities are such that we find it convenient to follow the terminology of [10] and also refer to our approximation as a PITC approximation.

We have already noted that our sparse approximation reduces the computational complexity of multi-output regression with GPs to that of applying independent GPs to each output. For larger data sets the $N^3$ term in the computational complexity and the $N^2$ term in the storage is still likely to be prohibitive. However, we can be inspired by the analogy of our approach to the PITC approximation and consider a more radical factorization of the outputs. In the fully independent training conditional (FITC) [13, 14] a factorization across the data points is assumed. For us that would lead to the following expression for conditional distribution of the output functions given the inducing variables, $p(\mathbf{y}|\mathbf{u}, \mathbf{Z}, \mathbf{X}, \boldsymbol{\theta}) = \prod_{q=1}^{Q}\prod_{n=1}^{N} p(y_{qn}|\mathbf{u}, \mathbf{Z}, \mathbf{X}, \boldsymbol{\theta})$ which can be briefly expressed through (6) with $\mathbf{D} = \mathrm{diag}\left[\mathbf{K_{f,f}} - \mathbf{K_{f,u}K_{u,u}^{-1}K_{u,f}}\right] = \left[\mathbf{K_{f,f}} - \mathbf{K_{f,u}K_{u,u}^{-1}K_{u,f}}\right] \odot \mathbf{M}$, with $\mathbf{M} = \mathbf{I}_Q \otimes \mathbf{I}_N$. Similar equations are obtained for the posterior (8), predictive (9) and marginal likelihood distributions (7) leading to the Fully Independent Training Conditional (FITC) approximation [13, 10]. Note that the marginal likelihood might be optimized both with respect to the parameters associated with the covariance matrices and with respect to $\mathbf{Z}$. In supplementary material we include the derivatives of the marginal likelihood wrt the matrices $\mathbf{K_{f,f}}$, $\mathbf{K_{u,f}}$ and $\mathbf{K_{u,u}}$.

## 4 Related work

There have been several suggestions for constructing multiple output GPs [2, 15, 1]. Under the convolution process framework, the semiparametric latent factor model (SLFM) proposed in [15] corresponds to a specific choice for the smoothing kernel function in (1) namely, $k_{qr}(\mathbf{x}) = \phi_{qr}\delta(\mathbf{x})$. The latent functions are assumed to be independent GPs and in such a case, $\mathrm{cov}\left[f_q(\mathbf{x}), f_s(\mathbf{x}')\right] = \sum_r \phi_{qr}\phi_{sr}k_{u_ru_r}(\mathbf{x}, \mathbf{x}')$. This can be written using matrix notation as $\mathbf{K_{f,f}} = (\boldsymbol{\Phi} \otimes \mathbf{I})\mathbf{K_{u,u}}(\boldsymbol{\Phi}^\top \otimes \mathbf{I})$. For computational speed up the informative vector machine (IVM) is employed [8].

In the multi-task learning model (MTLM) proposed in [1], the covariance matrix is expressed as $\mathbf{K_{f,f}} = K^f \otimes k(\mathbf{x}, \mathbf{x}')$, with $K^f$ being constrained positive semi-definite and $k(\mathbf{x}, \mathbf{x}')$ a covariance function over inputs. The Nyström approximation is applied to $k(\mathbf{x}, \mathbf{x}')$. As stated in [1] with respect to SLFM, the convolution process is related with MTLM when the smoothing kernel function is

given again by $k_{qr}(\mathbf{x}) = \phi_{qr}\delta(\mathbf{x})$ and there is only one latent function with covariance $k_{uu}(\mathbf{x}, \mathbf{x}') = k(\mathbf{x}, \mathbf{x}')$. In this way, $\mathrm{cov}\left[f_q(\mathbf{x}), f_s(\mathbf{x}')\right] = \phi_q\phi_s k(\mathbf{x}, \mathbf{x}')$ and in matrix notation $\mathbf{K_{f,f}} = \boldsymbol{\Phi}\boldsymbol{\Phi}^\top \otimes k(\mathbf{x}, \mathbf{x}')$. In [2], the latent processes correspond to white Gaussian noises and the covariance matrix is given by eq. (3). In this work, the complexity of the computational load is not discussed. Finally, [12] use a similar covariance function to the MTLM approach but use an IVM style approach to sparsification.

Note that in each of the approaches detailed above a $\delta$ function is introduced into the integral. In the dependent GP model of [2] it is introduced in the covariance function. Our approach considers the more general case when neither kernel nor covariance function is given by the $\delta$ function.

## 5   Results

For all our experiments we considered squared exponential covariance functions for the latent process of the form $k_{u_r u_r}(\mathbf{x}, \mathbf{x}') = \exp\left[-\frac{1}{2}\left(\mathbf{x} - \mathbf{x}'\right)^\top \mathbf{L}_r \left(\mathbf{x} - \mathbf{x}'\right)\right]$, where $\mathbf{L}_r$ is a diagonal matrix which allows for different length-scales along each dimension. The smoothing kernel had the same form, $k_{qr}(\boldsymbol{\tau}) = \frac{S_{qr}|\mathbf{L}_{qr}|^{1/2}}{(2\pi)^{p/2}}\exp\left[-\frac{1}{2}\boldsymbol{\tau}^\top \mathbf{L}_{qr}\boldsymbol{\tau}\right]$, where $S_{qr} \in \mathbb{R}$ and $\mathbf{L}_{qr}$ is a symmetric positive definite matrix. For this kernel/covariance function combination the necessary integrals are tractable (see supplementary material).

We first setup a toy problem in which we evaluate the quality of the prediction and the speed of the approximation. The toy problem consists of $Q = 4$ outputs, one latent function, $R = 1$, and $N = 200$ observation points for each output. The training data was sampled from the full GP with the following parameters, $S_{11} = S_{21} = 1$, $S_{31} = S_{41} = 5$, $L_{11} = L_{21} = 50$, $L_{31} = 300$, $L_{41} = 200$ for the outputs and $L_1 = 100$ for the latent function. For the independent processes, $w_q(\mathbf{x})$, we simply added white noise with variances $\sigma_1^2 = \sigma_2^2 = 0.0125$, $\sigma_3^2 = 1.2$ and $\sigma_4^2 = 1$. For the sparse approximations we used $M = 30$ fixed inducing points equally spaced between the range of the input and $R = 1$. We sought the kernel parameters through maximizing the marginal likelihood using a scaled conjugate gradient algorithm. For test data we removed a portion of one output as shown in Figure 1 (points in the interval $[-0.8, 0]$ were removed). The predictions shown correspond to the full GP (Figure 1(a)), an independent GP (Figure 1(b)), the FITC approximation (Figure 1(c)) and the PITC approximation (Figure 1(d)). Due to the strong dependencies between the signals, our model is able to capture the correlations and predicts accurately the missing information.

Table 1 shows prediction results over an independent test set. We used 300 points to compute the standarized mean square error (SMSE) [11] and ten repetitions of the experiment, so that we also included one standard deviation for the ten repetitions. The training times for iteration of each model are $1.45 \pm 0.23$ secs for the full GP, $0.29 \pm 0.02$ secs for the FITC and $0.48 \pm 0.01$ for the PITC. Table 1, shows that the SMSE of the sparse approximations is similar to the one obtained with the full GP with a considerable reduction of training times.

| Method | Output 1 | Output 2 | Output 3 | Output 4 |
|--------|----------|----------|----------|----------|
| Full GP | $1.07 \pm 0.08$ | $0.99 \pm 0.03$ | $1.12 \pm 0.07$ | $1.05 \pm 0.07$ |
| FITC | $1.08 \pm 0.09$ | $1.00 \pm 0.03$ | $1.13 \pm 0.07$ | $1.04 \pm 0.07$ |
| PITC | $1.07 \pm 0.08$ | $0.99 \pm 0.03$ | $1.12 \pm 0.07$ | $1.05 \pm 0.07$ |

Table 1: Standarized mean square error (SMSE) for the toy problem over an independent test set. All numbers are to be multiplied by $10^{-2}$. The experiment was repeated ten times. Table included the value of one standard deviation over the ten repetitions.

We now follow a similar analysis for a dataset consisting of weather data collected from a sensor network located on the south coast of England. The network includes four sensors (named Bramblemet, Sotonmet, Cambermet and Chimet) each of which measures several environmental variables [12]. We selected one of the sensors signals, tide height, and applied the PITC approximation scheme with an additional *squared exponential* independent kernel for each $w_q(\mathbf{x})$ [11]. Here $Q = 4$ and we chose $N = 1000$ of the 4320 for the training set, leaving the remaining points for testing. For comparison we also trained a set of independent GP models. We followed [12] in simulating sensor failure by introducing some missing ranges for these signals. In particular, we have a missing range

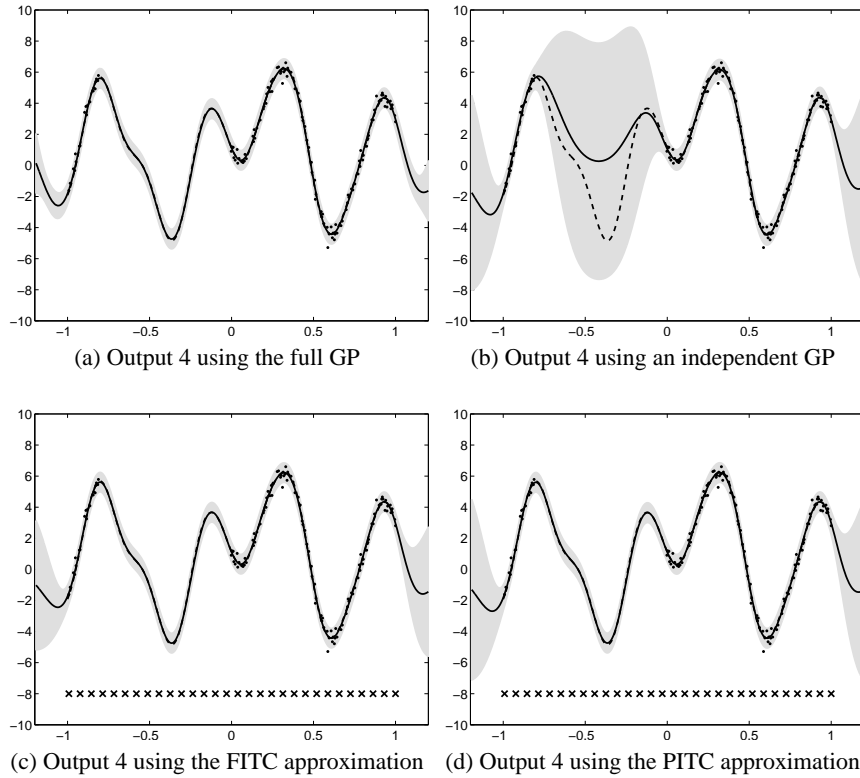

(a) Output 4 using the full GP     (b) Output 4 using an independent GP

(c) Output 4 using the FITC approximation     (d) Output 4 using the PITC approximation

Figure 1: Predictive mean and variance using the full multi-output GP, the sparse approximation and an independent GP for output 4. The solid line corresponds to the mean predictive, the shaded region corresponds to 2 standard deviations away from the mean and the dash line is the actual value of the signal without noise. The dots are the noisy training points. There is a range of missing data in the interval $[-0.8, 0.0]$. The crosses in figures 1(c) and 1(d) corresponds to the locations of the inducing inputs.

of $[0.6, 1.2]$ for the Bramblemet tide height sensor and $[1.5, 2.1]$ for the Cambermet. For the other two sensors we used all $1000$ training observations. For the sparse approximation we took $M = 100$ equally spaced inducing inputs. We see from Figure 2 that the PITC approximation captures the dependencies and predicts closely the behavior of the signal in the missing range. This contrasts with the behavior of the independent model, which is not able to follow the original signal.

As another example we employ the Jura dataset, which consists of measurements of concentrations of several heavy metals collected in the topsoil of a $14.5$ km$^2$ region of the Swiss Jura. The data is divided into a prediction set ($259$ locations) and a validation set ($100$ locations)[1]. In a typical situation, referred as *undersampled* or *heterotopic* case, a few expensive measurements of the attribute of interest are supplemented by more abundant data on correlated attributes that are cheaper to sample. We follow the experiments described in [5, p. 248,249] in which a *primary variable* (cadmium and copper) at prediction locations in conjunction with some *secondary variables* (nickel and zinc for cadmium; lead, nickel and zinc for copper) at prediction and validation locations, are employed to predict the concentration of the primary variable at validation locations. We compare results of independent GP, the PITC approximation, the full GP and ordinary co-kriging. For the PITC experiments, a *k-means* procedure is employed first to find the initial locations of the inducing values and then these locations are optimized in the same optimization procedure used for the parameters. Each experiment is repeated ten times. The results for ordinary co-kriging were obtained from [5, p. 248,249]. In this case, no values for standard deviation are reported. Figure 3 shows results of prediction for cadmium (Cd) and copper (Cu). From figure 3(a), it can be noticed that using 50 inducing values, the approximation exhibits a similar performance to the co-kriging method. As more

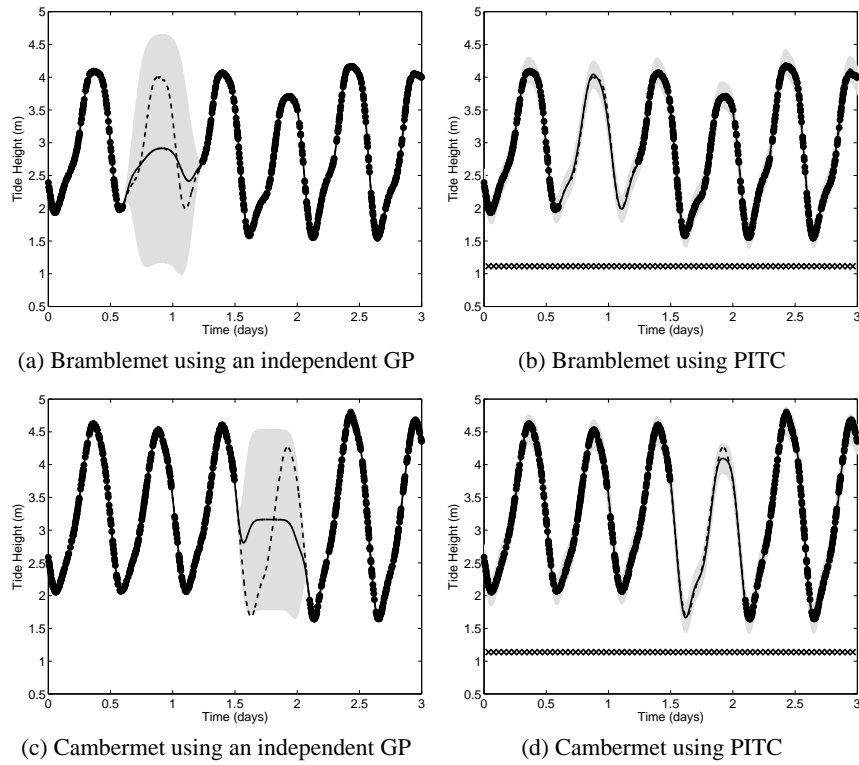

(a) Bramblemet using an independent GP    (b) Bramblemet using PITC

(c) Cambermet using an independent GP    (d) Cambermet using PITC

Figure 2: Predictive Mean and variance using independent GPs and the PITC approximation for the tide height signal in the sensor dataset. The dots indicate the training observations while the dash indicates the testing observations. We have emphasized the size of the training points to differentiate them from the testing points. The solid line corresponds to the mean predictive. The crosses in figures 2(b) and 2(d) corresponds to the locations of the inducing inputs.

inducing values are included, the approximation follows the performance of the full GP, as it would be expected. From figure 3(b), it can be observed that, although the approximation is better that the independent GP, it does not obtain similar results to the full GP. Summary statistics of the prediction data ([5, p. 15]) shows higher variability for the copper dataset than for the cadmium dataset, which explains in some extent the different behaviors.

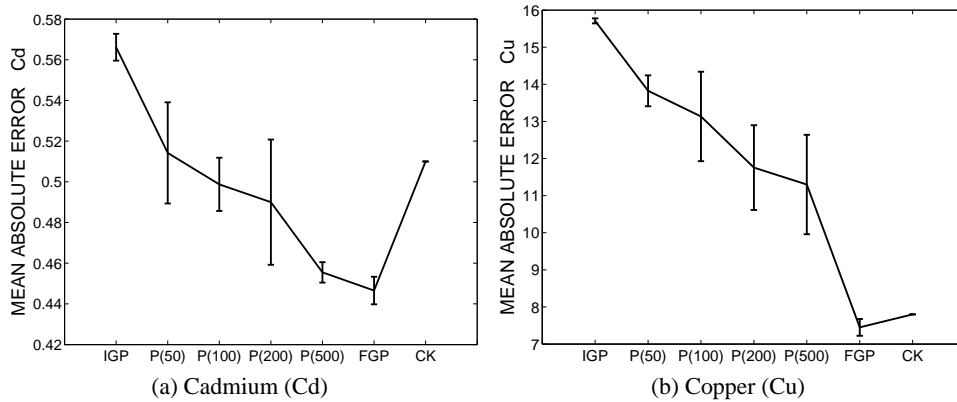

(a) Cadmium (Cd)    (b) Copper (Cu)

Figure 3: Mean absolute error and standard deviation for ten repetitions of the experiment for the Jura dataset In the bottom of each figure, IGP stands for independent GP, P($M$) stands for PITC with $M$ inducing values, FGP stands for full GP and CK stands for ordinary co-kriging (see [5] for detailed description).

# 6 Conclusions

We have presented a sparse approximation for multiple output GPs, capturing the correlated information among outputs and reducing the amount of computational load for prediction and optimization purposes. The reduction in computational complexity for the PITC approximation is from $O(N^3Q^3)$ to $O(N^3Q)$. This matches the computational complexity for modeling with independent GPs. However, as we have seen, the predictive power of independent GPs is lower.

Linear dynamical systems responses can be expressed as a convolution between the impulse response of the system with some input function. This convolution approach is an equivalent way of representing the behavior of the system through a linear differential equation. For systems involving high amounts of coupled differential equations [4], the approach presented here is a reasonable way of obtaining approximate solutions and incorporating prior domain knowledge to the model.

One could optimize with respect to positions of the values of the latent functions. As the input dimension grows, it might be more difficult to obtain an acceptable response. Some solutions to this problem have already been proposed [14].

## Acknowledgments

We thank the authors of [12] who kindly made the sensor network database available.

## Footnotes

[1]This data is available at http://www.ai-geostats.org/

## References

[1] E. V. Bonilla, K. M. Chai, and C. K. I. Williams. Multi-task Gaussian process prediction. In J. C. Platt, D. Koller, Y. Singer, and S. Roweis, editors, *NIPS*, volume 20, Cambridge, MA, 2008. MIT Press. In press.

[2] P. Boyle and M. Frean. Dependent Gaussian processes. In L. Saul, Y. Weiss, and L. Bouttou, editors, *NIPS*, volume 17, pages 217–224, Cambridge, MA, 2005. MIT Press.

[3] M. Brookes. The matrix reference manual. Available on-line., 2005. http://www.ee.ic.ac.uk/hp/staff/dmb/matrix/intro.html.

[4] P. Gao, A. Honkela, M. Rattray, and N. D. Lawrence. Gaussian process modelling of latent chemical species: Applications to inferring transcription factor activities. *Bioinformatics*, 24(16):i70–i75, 2008.

[5] P. Goovaerts. *Geostatistics For Natural Resources Evaluation*. Oxford University Press, 1997. ISBN 0-19-511538-4.

[6] D. M. Higdon. Space and space-time modelling using process convolutions. In C. Anderson, V. Barnett, P. Chatwin, and A. El-Shaarawi, editors, *Quantitative methods for current environmental issues*, pages 37–56. Springer-Verlag, 2002.

[7] N. D. Lawrence. Learning for larger datasets with the Gaussian process latent variable model. In Meila and Shen [9].

[8] N. D. Lawrence, M. Seeger, and R. Herbrich. Fast sparse Gaussian process methods: The informative vector machine. In S. Becker, S. Thrun, and K. Obermayer, editors, *NIPS*, volume 15, pages 625–632, Cambridge, MA, 2003. MIT Press.

[9] M. Meila and X. Shen, editors. *AISTATS*, San Juan, Puerto Rico, 21-24 March 2007. Omnipress.

[10] J. Quiñonero Candela and C. E. Rasmussen. A unifying view of sparse approximate Gaussian process regression. *JMLR*, 6:1939–1959, 2005.

[11] C. E. Rasmussen and C. K. I. Williams. *Gaussian Processes for Machine Learning*. MIT Press, Cambridge, MA, 2006. ISBN 0-262-18253-X.

[12] A. Rogers, M. A. Osborne, S. D. Ramchurn, S. J. Roberts, and N. R. Jennings. Towards real-time information processing of sensor network data using computationally efficient multi-output Gaussian processes. In *Proceedings of the International Conference on Information Processing in Sensor Networks (IPSN 2008)*, 2008. In press.

[13] E. Snelson and Z. Ghahramani. Sparse Gaussian processes using pseudo-inputs. In Y. Weiss, B. Schölkopf, and J. C. Platt, editors, *NIPS*, volume 18, Cambridge, MA, 2006. MIT Press.

[14] E. Snelson and Z. Ghahramani. Local and global sparse Gaussian process approximations. In Meila and Shen [9].

[15] Y. W. Teh, M. Seeger, and M. I. Jordan. Semiparametric latent factor models. In R. G. Cowell and Z. Ghahramani, editors, *AISTATS 10*, pages 333–340, Barbados, 6-8 January 2005. Society for Artificial Intelligence and Statistics.
